# A General Purpose Image Processing Chip: Orientation Detection

**Ralph Etienne-Cummings and Donghui Cai**
Department of Electrical Engineering
Southern Illinois University
Carbondale, IL 62901-6603

## Abstract

A 80 x 78 pixel general purpose vision chip for spatial focal plane processing is presented. The size and configuration of the processing receptive field are programmable. The chip's architecture allows the photoreceptor cells to be small and densely packed by performing all computation on the read-out, away from the array. In addition to the raw intensity image, the chip outputs four processed images in parallel. Also presented is an application of the chip to line segment orientation detection, as found in the retinal receptive fields of toads.

## 1 INTRODUCTION

The front-end of the biological vision system is the retina, which is a layered structure responsible for image acquisition and pre-processing. The early processing is used to extract spatiotemporal information which helps perception and survival. This is accomplished with cells having feature detecting receptive fields, such as the edge detecting center-surround spatial receptive fields of the primate and cat bipolar cells [Spillmann, 1990]. In toads, the receptive fields of the retinal cells are even more specialized for survival by detecting "prey" and "predator" (from size and orientation filters) at this very early stage [Spillmann, 1990].

The receptive of the retinal cells performs a convolution with the incident image in parallel and continuous time. This has inspired many engineers to develop retinomorphic vision systems which also imitate these parallel processing capabilities [Mead, 1989; Camp, 1994]. While this approach is ideal for fast early processing, it is not space efficient. That is, in realizing the receptive field within each pixel, considerable die area is required to implement the convolution kernel. In addition, should programmability be required, the complexity of each pixel increases drastically. The space constraints are eliminated if the processing is performed serially during read-out. The benefits of this approach are 1) each pixel can be as small as possible to allow high resolution imaging, 2) a single processor unit is used for the entire retina thus reducing mis-match problems, 3) programmability can be obtained with no impact on the density of imaging array, and

4) compact general purpose focal plane visual processing is realizable. The space constrains are then transformed into temporal restrictions since the scanning clock speed and response time of the processing circuits must scale with the size of the array. Dividing the array into sub-arrays which are scanned in parallel can help this problem. Clearly this approach departs from the architecture of its biological counterpart, however, this method capitalizes on the main advantage of silicon which is its speed. This is an example of mixed signal neuromorphic engineering, where biological ideas are mapped onto silicon not using direct imitation (which has been the preferred approach in the past) but rather by realizing their *essence* with the best silicon architecture and computational circuits.

This paper presents a general purpose vision chip for spatial focal plane processing. Its architecture allows the photoreceptor cells to be small and densely packed by performing all computation on the read-out, away from the array. Performing computation during read-out is ideal for silicon implementation since no additional temporal over-head is required, provided that the processing circuits are fast enough. The chip uses a single convolution kernel, per parallel sub-array, and the scanning bit pattern to realize various receptive fields. This is different from other focal plane image processors which are usually restricted to hardwired convolution kernels, such as oriented 2D Gabor filters [Camp, 1994]. In addition to the raw intensity image, the chip outputs four processed versions per sub-array. Also presented is an application of the chip to line segment orientation detection, as found in the retinal receptive fields of toads [Spillmann, 1990].

## 2 THE GENERAL PURPOSE IMAGE PROCESSING CHIP

### 2.1 System Overview

This chip has an 80 row by 78 column photocell array partitioned into four independent sub-arrays, which are scanned and output in parallel, (see figure 1). Each block is 40 row by 39 column, and has its own convolution kernel and output circuit. The scanning circuit includes three parts: virtual ground, control signal generator (CSG), and scanning output transformer. Each block has its own virtual ground and scanning output transformer in both x direction (horizontal) and y direction (vertical). The control signal generator is shared among blocks.

### 2.2 Hardware Implementation

The photocell is composed of phototransistor, photo current amplifier, and output control. The phototransistor performance light transduction, while the amplifier magnifies the photocurrent by three orders of magnitude. The output control provides multiple copies of the amplified photocurrent which is subsequently used for focal plane image processing.

The phototransistor is a parasitic PNP transistor in an Nwell CMOS process. The current amplifier uses a pair of diode connected pmosfets to obtain a logarithmic relationship between light intensity and output current. This circuit also amplifies the photocurrent from nanoamperes to microamperes. The photocell sends three copies of the output currents into three independent buses. The connections from the photocell to the buses are controlled by pass transistors, as shown in Fig. 2. The three current outputs allow the image to be processed using multiple receptive field organization (convolution kernels), while the raw image is also output. The row (column) buses provides currents for extracting horizontally (vertically) oriented image features, while the original bus provides the logarithmically compressed intensity image.

The scanning circuit addresses the photocell array by selecting groups of cells at one time. Since the output of the cells are currents, virtual ground circuits are used on each bus to mask the > 1pF capacitance of the buses. The CSG, implemented with shift registers

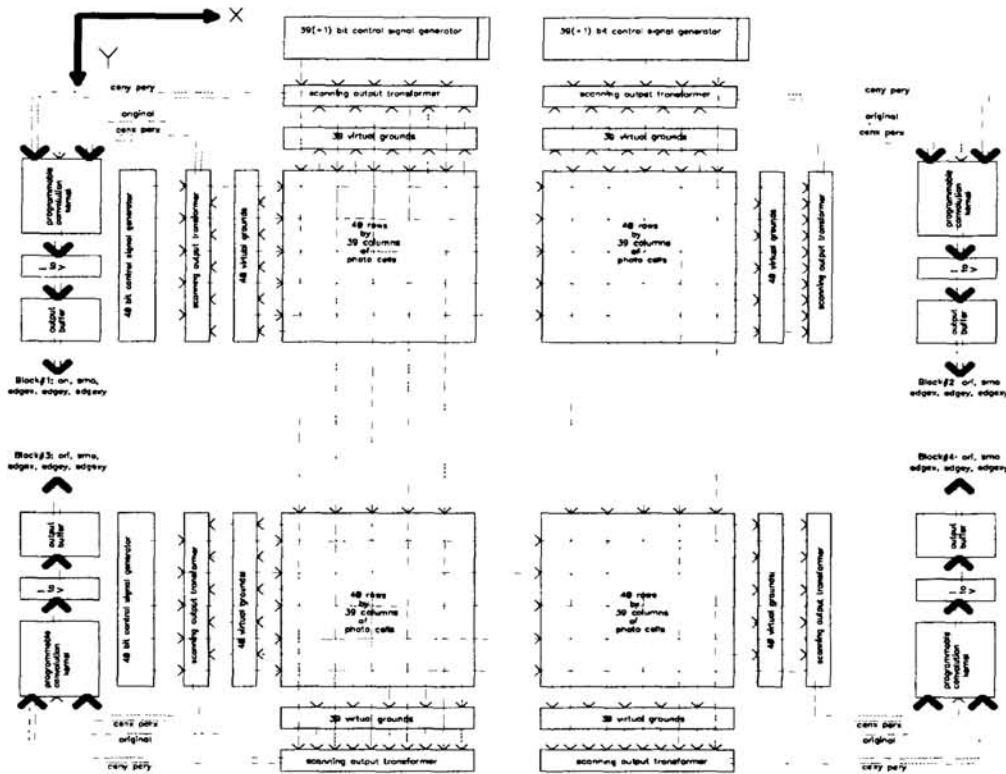

Figure 1:  Block diagram of the chip.

produces signals which select photocells and control the scanning output transformer. The scanning output transformer converts currents from all row buses into $I_{perx}$ and $I_{cenx}$, and converts currents from all row buses into $I_{pery}$ and $I_{ceny}$.   This transformation is required to implement the various convolution kernels discussed later.

The output transformer circuits are controlled by a central CSG and a peripheral CSG. These two generators have identical structures but different initial values.  It consists of an n-bit shift register in x direction (horizontally) and an m-bit shift register in y direction (vertically).   A feedback circuit is used to restore the scanning pattern into the x shift register after each row scan is completed.  This is repeated until all the row in each block are scanned.

The control signals from the peripheral and central  CSGs select all the cells covered by a 2D convolution mask (receptive field).  The selected cells send $I_{xy}$ to the original bus, $I_{xp}$ to the row bus, and $I_{yp}$ to the column bus.   The function of the scanning output transformer is to identify which rows (columns) are considered as the center ($I_{cenx}$ or $I_{ceny}$) or periphery ($I_{perx}$ or $I_{pery}$) of the convolution kernel, respectively.  Figure 3 shows how a 3x3 convolution kernel can be constructed.

Figure 4 shows how the output transformer works for a 3x3 mask.   Only row bus transformation is shown in this example, but the same mechanism applies to the column bus as well.  The photocell array is m row by n column, and the size is 3x3.  The XC (x center) and YC (y center) come from the central CSG;  while XP (x peripheral) and YP (y peripheral) come from the peripheral CSG.   After loading the CSG, the initial values of XP and YP are both 00011...1.   The initial values of XC and YC are both 10111...1. This identifies the central cell as location (2, 2).   The currents from the central row (column) are summed to form $I_{cenx}$ and $I_{ceny}$, while all the peripheral cells are summed to form $I_{perx}$ and $I_{pery}$. This is achieved by activating the switches labeled XC, YC, XP and YP in figure 2.  $XP_i$ ($YP_i$) {i=1, 2, ..., n} controls whether the output current of one cell

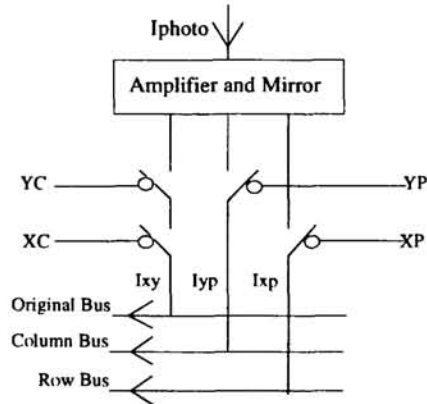 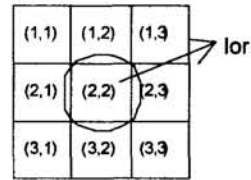 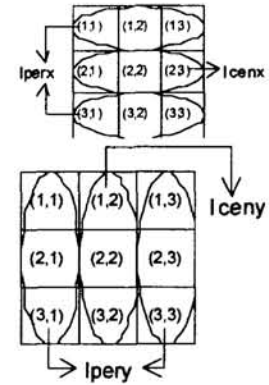

Figure 2: Connections between a photo-cell and the current buses.

Figure 3: Constructing a 3x3 receptive field.

goes to the row (column) bus. Since $XP_i$ ($YP_i$) is connected to the gate of a pmos switch, a 0 in $XP_i$ ($YP_i$) it turns on. $YC_i$ ($XC_i$) {i=1, 2, ..., n} controls whether a row (column) bus connects to $I_{cenx}$ bus in the same way. On the other hand, the connection from a row (column) bus to $I_{perx}$ bus is controlled by an nmos and a pmos switch. The connection is made if and only if $YC_i$ ($XC_i$), an nmos switch, is 1 and $YP_i$ ($XP_i$), a pmos switches, is 0. The intensity image is obtained directly when $XC_i$ and $YC_i$ are both 0. Hence, $I_{ori} = I(2,2)$, $I_{cenx} = I_{row2} = I(2,1) + I(2,2) + I(2,3)$ and $I_{perx} = I_{row1} + I_{row3} = I(1,1) + I(1,2) + I(1,3) + I(3,1) + I(3,2) + I(3,3)$.

The convolution kernel can be programmed to perform many image processing tasks by loading the scanning circuit with the appropriate bit pattern. This is illustrated by configuring the chip to perform image smoothing and edge extraction (x edge, y edge, and 2D edge), which are all computed simultaneously on read-out. It receives five inputs ($I_{ori}$, $I_{cenx}$, $I_{perx}$, $I_{ceny}$, $I_{pery}$) from the scanning circuit and produces five outputs ($I_{ori}$, $I_{edgex}$, $I_{edgey}$, $I_{smooth}$, $I_{edge2d}$). The kernel (receptive field) size is programmable from 3x3, 5x5, 7x7, 9x9 and 11x11. Fig. 5 shows the 3x3 masks for this processing. Repeating the above steps for 5x5, 7x7, 9x9, and 11x11 masks, we can get similar results.

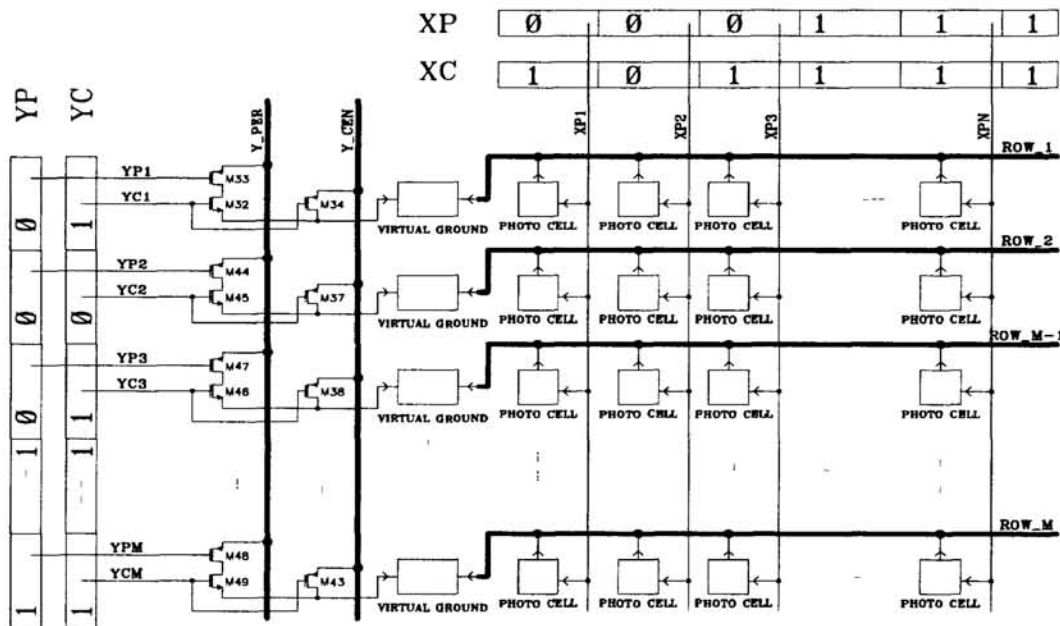

Figure 4: Scanning output transformer for an m row by n column photo cell array.

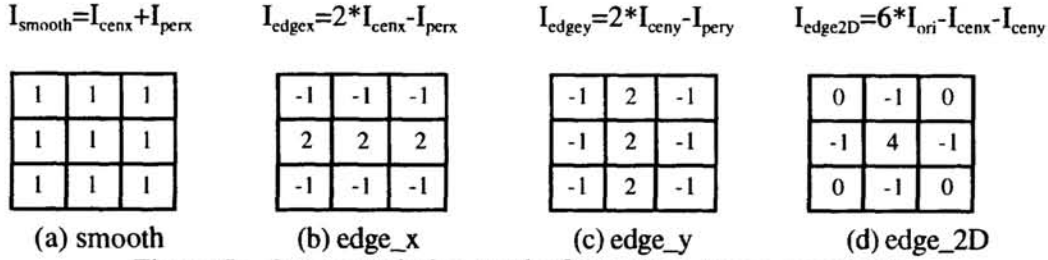

$I_{smooth}=I_{cenx}+I_{perx}$    $I_{edgex}=2*I_{cenx}-I_{perx}$    $I_{edgey}=2*I_{ceny}-I_{pery}$    $I_{edge2D}=6*I_{ori}-I_{cenx}-I_{ceny}$

| (a) smooth | (b) edge_x | (c) edge_y | (d) edge_2D |

Figure 5: 3x3 convolution masks for various image processing.

In general, the convolution results under different mask sizes can be expressed as follows:

$I_{smooth}=I_{cenx}+I_{perx}$    $I_{edgex}=K_{1d}*I_{cenx}-I_{perx}$    $I_{edgey}=K_{1d}*I_{ceny}-I_{pery}$    $I_{edge2D}=K_{2d}*I_{ori}-I_{cenx}-I_{ceny}$

Where $K_{1d}$ and $K_{2d}$ are the programmable coefficients (from 2-6 and 6-14, respectively) for 1D edge extraction and 2D edge extraction, respectively. By varying the locations of the 0's in the scanning circuits, different types of receptive fields (convolution kernels) can be realized.

## 2.3  Results

The chip contains 65K transistors in a footprint of 4.6 mm x 4.7 mm. There are 80 x 78 photocells in the chip, each of which is 45.6 µm x 45 µm and a fill factor of 15%. The convolution kernel occupies 690.6 µm x 102.6 µm. The power consumption of the chip for a 3x3 (11x11) receptive field, indoor light, and 5V power supply is < 2 mW (8 mW).

To capitalize on the programmability of this chip, an A/D card in a Pentium 133MHz PC is used to load the scanning circuit and to collect data. The card, which has a maximum analog throughput of 100 KHz limits the frame rate of the chip to 12 frames per second. At this rate, five processed versions of the image is collected and displayed. The scanning and processing circuits can operate at 10 MHz (6250 fps), however, the phototransistors have much slower dynamics. Temporal smoothing (smear) can be observed on the scope when the frame rate exceeds 100 fps.

The chip displays a logarithmic relationship between light intensity and output current (unprocessed imaged) from 0.1 lux (100 nA) to 6000 lux (10 µA). The fixed pattern noise, defined as standard-deviation/mean, decreases abruptly from 25% in the dark to 2% at room light (800 lux). This behavior is expected since the variation of individual pixel current is large compared to the mean output when the mean is small. The logarithmic response of the photocell results in high sensitivity at low light, thus increasing the mean value sharply. Little variation is observed between chips.

The contrast sensitivity of the edge detection masks is also measured for the 3x3 and 5x5 receptive fields. Here contrast is defined as $(I_{max} - I_{min})/(I_{max} + I_{min})$ and sensitivity is given as a percentage of the maximum output. The measurements are performed for normal room and bright lighting conditions. Since the two conditions corresponded to the saturated part of the logarithmic transfer function of the photocells, then a linear relationship between output response and contrast is expected. Figure 6 shows contrast sensitivity plot. Figure 7 shows examples of chip's outputs. The top two images are the raw and smoothed (5x5) images. The bottom two are the 1D edge_x (left) and 2D edge (right) images. The pixels with positive values have been thresholded to white. The vertical black line in the image is not visible in the edge_x image, but can be clearly seen in the edge_2D image.

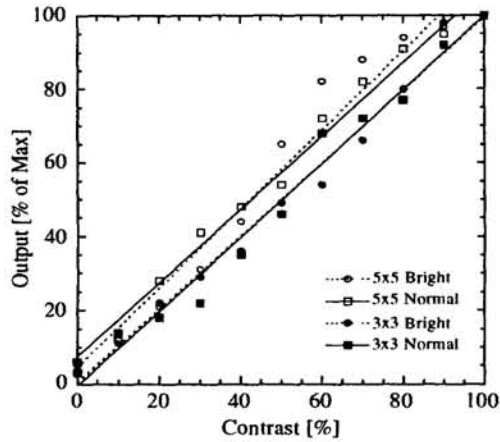

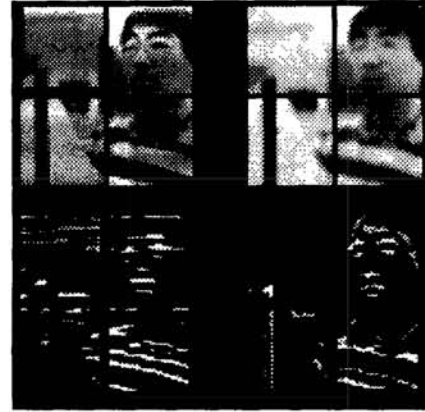

Figure 6: Contrast sensitivity function of the x edge detection mask.

Figure 7: (Clockwise) Raw image, 5x5 smoothed image, edge_2D and edge_x.

# 3 APPLICATION: ORIENTATION DETECTION

## 3.1 Algorithm Overview

This vision chip can be elegantly used to measure the orientation of line segments which fall across the receptive field of each pixel. The output of the 1D Laplacian operators, edge_x and edge_y, shown in figure 5, can be used to determine the orientation of edge segments. Consider a continuous line through the origin, represented by a delta function in 2D space by $\delta(y-x\tan\theta)$. If the origin is the center of the receptive field, the response of the edge_x kernel can be computed by evaluating the convolution equation (1), where $W(x) = u(x+m)-u(x-m)$ is the x window over which smoothing is performed, $2m+1$ is the width of the window and $2n+1$ is the number of coefficients realizing the discrete Laplacian operator. In our case, $n = m$. Evaluating this equation and substituting the origin for the pixel location yields equation (2), which indicates that the output of the 1D edge_x (edge_y) detectors have a discretized linear relationship to orientation from $0°$ to $45°$ ($45°$ to $90°$). At $0°$, the second term in equation (2) is zero. As $\theta$ increase, more terms are subtracted until all terms are subtracted at $45°$. Above $45°$ (below $45°$), the edge_x (edge_y) detectors output zero since equal numbers of positive and negative coefficients are summed. Provided that contrast can be normalized, the output of the detectors can be used to extract the orientation of the line. Clearly these responses are even about the x- and y-axis, respectively. Hence, a second pair of edge detectors, oriented at $45°$, is required to uniquely extract the angle of the line segment.

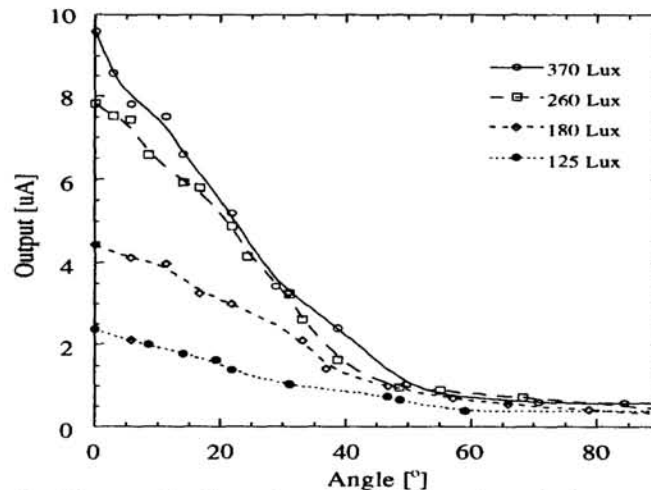

Figure 8: Measured orientation transfer function of edge_x detectors.

$$O_{edge\_x}(x,y) = [\,2nW(x \pm m)\delta(y) - \sum_{i=1}^{n} W(x \pm m)\delta(y \pm i)\,] * \delta(y - x\tan\theta) \qquad (1)$$

$$O_{edge\_x}(0,0) = 2n - [\,\sum_{i=1}^{n}(W(\frac{i}{\tan\theta}) + W(\frac{-i}{\tan\theta}))\,] \qquad (2)$$

## 3.2   Results

Figure 8 shows the measured output of the edge_x detectors for various lighting conditions as a line is rotated. The average positive outputs are plotted. As expected, the output is maximum for bright ambients when the line is horizontal. As the line is rotated, the output current decreases linearly and levels off at approximately 45°. On the other hand, the edge_y (not shown) begins its linear increase at 45° and maximizes at 90°. After normalizing for brightness, the four curves are very similar (not shown).

To further demonstrate orientation detection with this chip, a character consisting of a circle and some straight lines is presented. The intensity image of the character is shown in figure 9(a). Figures 9(b) and 9(c) show the outputs of the edge_x and edge_y detectors, respectively. Since a 7x7 receptive field is used in this experiment, some outer pixels of each block are lost. The orientation selectivity of the 1D edge detectors are clearly visible in the figures, where edge_x highlights horizontal edges and edge_y vertical edges. Figure 9(d) shows the reported angles. A program is written which takes the two 1D edge images, finds the location of the edges from the edge_2D image, the intensity at the edges (positive lobe) and then computes the angle of the edge segment. In figure 9(d), the black background is chosen for locations where no edges are detected, white is used for 0° and gray for 90°.

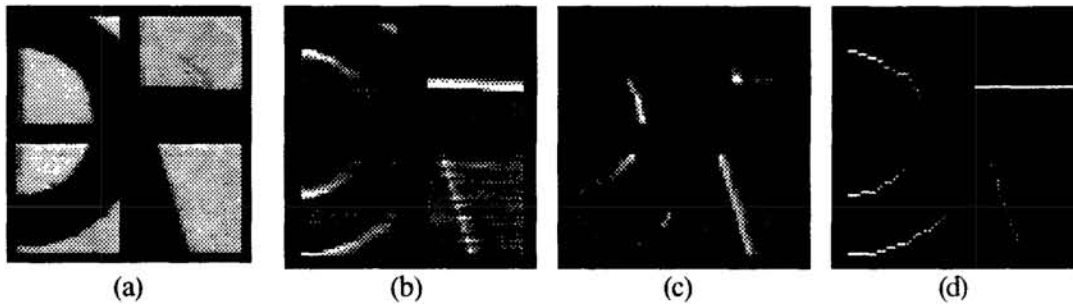

(a)              (b)              (c)              (d)

Figure 9:  Orientation detection using 1D Laplacian Operators.

## 4   CONCLUSION

A 80x78 pixel general purpose vision chip for spatial focal plane processing has been presented. The size and configuration of the processing receptive field are programmable. In addition to the raw intensity image, the chip outputs four processed images in parallel. The chip has been successfully used for compact line segment orientation detection, which can be used in character recognition. The programmability and relatively low power consumption makes it ideal for many visual processing tasks.

## References

Camp W. and J. Van der Spiegel, "A Silicon VLSI Optical Sensor for Pattern Recognition, " *Sensors and Actuators A*, Vol. 43, No. 1-3, pp. 188-195, 1994.

Mead C. and M. Ismail (Eds.), *Analog VLSI Implementation of Neural Networks*, Kluwer Academic Press, Newell, MA, 1989.

Spillmann L. and J. Werner (Eds.), *Visual Perception: The Neurophysiological Foundations*, Academic Press, San Diego, CA, 1990.